# A model of the hippocampus combining self-organization and associative memory function.

Michael E. Hasselmo, Eric Schnell
Joshua Berke and Edi Barkai
Dept. of Psychology, Harvard University
33 Kirkland St., Cambridge, MA 02138
hasselmo@katla.harvard.edu

## Abstract

A model of the hippocampus is presented which forms rapid self-organized representations of input arriving via the perforant path, performs recall of previous associations in region CA3, and performs comparison of this recall with afferent input in region CA1. This comparison drives feedback regulation of cholinergic modulation to set appropriate dynamics for learning of new representations in region CA3 and CA1. The network responds to novel patterns with increased cholinergic modulation, allowing storage of new self-organized representations, but responds to familiar patterns with a decrease in acetylcholine, allowing recall based on previous representations. This requires selectivity of the cholinergic suppression of synaptic transmission in stratum radiatum of regions CA3 and CA1, which has been demonstrated experimentally.

## 1    INTRODUCTION

A number of models of hippocampal function have been developed (Burgess et al., 1994; Myers and Gluck, 1994; Touretzky et al., 1994), but remarkably few simulations have addressed hippocampal function within the constraints provided by physiological and ana­tomical data. Theories of the function of specific subregions of the hippocampal forma­tion often do not address physiological mechanisms for changing dynamics between learning of novel stimuli and recall of familiar stimuli. For example, the afferent input to the hippocampus has been proposed to form orthogonal representations of entorhinal activity (Marr, 1971; McNaughton and Morris, 1987; Eichenbaum and Buckingham, 1990), but simulations have not addressed the problem of when these representations

should remain stable, and when they should be altered. In addition, models of autoassociative memory function in region CA3 (Marr, 1971; McNaughton and Morris, 1987; Levy, 1989; Eichenbaum and Buckingham, 1990) and heteroassociative memory function at the Schaffer collaterals projecting from region CA3 to CA1 (Levy, 1989; McNaughton, 1991) require very different activation dynamics during learning versus recall.

Acetylcholine may set appropriate dynamics for storing new information in the cortex (Hasselmo et al., 1992, 1993; Hasselmo, 1993, 1994; Hasselmo and Bower, 1993). Acetylcholine has been shown to selectively suppress synaptic transmission at intrinsic but not afferent fiber synapses (Hasselmo and Bower, 1992), to suppress the neuronal adaptation of cortical pyramidal cells (Hasselmo et al., 1994; Barkai and Hasselmo, 1994), and to enhance long-term potentiation of synaptic potentials (Hasselmo, 1994b). Models show that suppression of synaptic transmission during learning prevents recall of previously stored information from interfering with the storage of new information (Hasselmo et al., 1992, 1993; Hasselmo, 1993, 1994a), while cholinergic enhancement of synaptic modification enhances the rate of learning (Hasselmo, 1994b).

Feedback regulation of cholinergic modulation may set the appropriate level of cholinergic modulation dependent upon the novelty or familiarity of a particular input pattern. We have explored possible mechanisms for the feedback regulation of cholinergic modulation in simulations of region CA1 (Hasselmo and Schnell, 1994) and region CA3. Here we show that self-regulated learning and recall of self-organized representations can be obtained in a network simulation of the hippocampal formation. This model utilizes selective cholinergic suppression of synaptic transmission in stratum radiatum of region CA3, which has been demonstrated in brain slice preparations of the hippocampus.

## 2 METHODS

### 2.1. SIMPLIFIED REPRESENTATION OF HIPPOCAMPAL NEURONS.

In place of the sigmoid input-output functions used in many models, this model uses a simple representation in which the output of a neuron is not explicitly constrained, but the total network activity is regulated by feedback from inhibitory interneurons and adaptation due to intracellular calcium concentration. Separate variables represent pyramidal cell membrane potential $a$, intracellular calcium concentration $c$, and the membrane potential of inhibitory interneurons $h$:

$$\Delta a_i = A_i - \eta a_i - \mu c + \sum_j W_{ij} g(a_j - \theta_o) - H_{ik} g(h_k - \theta_h)$$

$$\Delta c_i = \gamma g(a_i - \theta_c) - \Omega c$$

$$\Delta h_k = \sum_j W_{kj} g(a_j - \theta_o) - \eta h_k - \sum_l H_{kl} g(h_l - \theta_o)$$

where A = afferent input, $\eta$ = passive decay of membrane potential, $\mu$ = strength of cal-

cium-dependent potassium current (proportional to intracellular calcium), $W_{ij}$ = excitatory recurrent synapses (longitudinal association path terminating in stratum radiatum), $g()$ is a threshold linear function proportional to the amount by which membrane potential exceeds an output threshold $\theta_o$ or threshold for calcium current $\theta_c$, $\gamma$ = strength of voltage-dependent calcium current, $\Omega$ = diffusion constant of calcium, $W_{ki}$ = excitatory synapses inhibitory interneurons, $H_{ik}$ = inhibitory synapses from interneurons to pyramidal cells, $H_{kl}$= inhibitory synapses between interneurons. This representation gives neurons adaptation characteristics similar to those observed with intracellular recording (Barkai and Hasselmo, 1994), including a prominent afterhyperpolarization potential (see Figure 1).

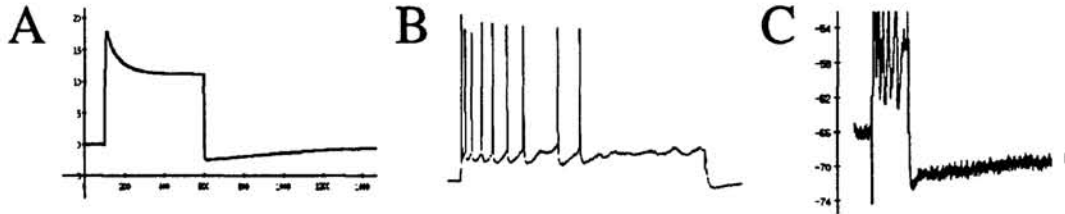

Figure 1. Comparison of pyramidal cell model with experimental data.

In Figure 1, A shows the membrane potential of a modeled pyramidal cell in response to simulated current injection. Output of this model is a continuous variable proportional to how much membrane potential exceeds threshold. This is analogous to the reciprocal of interspike interval in real neuronal recordings. Note that the model displays adaptation during current injection and afterhyperpolarization afterwards, due to the calcium-dependent potassium current. B shows the intracellularly recorded membrane potential in a piriform cortex pyramidal cell, demonstrating adaptation of firing frequency due to activation of calcium-dependent potassium current. The firing rate falls off in a manner similar to the smooth decrease in firing rate in the simplified representation. C shows an intracellular recording illustrating long-term afterhyperpolarization caused by calcium influx induced by spiking of the neuron during current injection.

## 2.2. NETWORK CONNECTIVITY

A schematic representation of the network simulation of the hippocampal formation is shown in Figure 2. The anatomy of the hippocampal formation is summarized on the left in A, and the function of these different subregions in the model is shown on the right in B. Each of the subregions in the model contained a population of excitatory neurons with a single inhibitory interneuron mediating feedback inhibition and keeping excitatory activity bounded. Thus, the local activation dynamics in each region follow the equations presented above. The connectivity of the network is further summarized in Figure 3 in the Results section. A learning rule of the Hebbian type was utilized at all synaptic connections, with the exception of the mossy fibers from the dentate gyrus to region CA3, and the connections to and from the medial septum. Self-organization of perforant path synapses was obtained through decay of synapses with only pre or post-synaptic activity, and growth of synapses with combined activity. Associative memory function at synapses

arising from region CA3 was obtained through synaptic modification during cholinergic suppression of synaptic transmission.

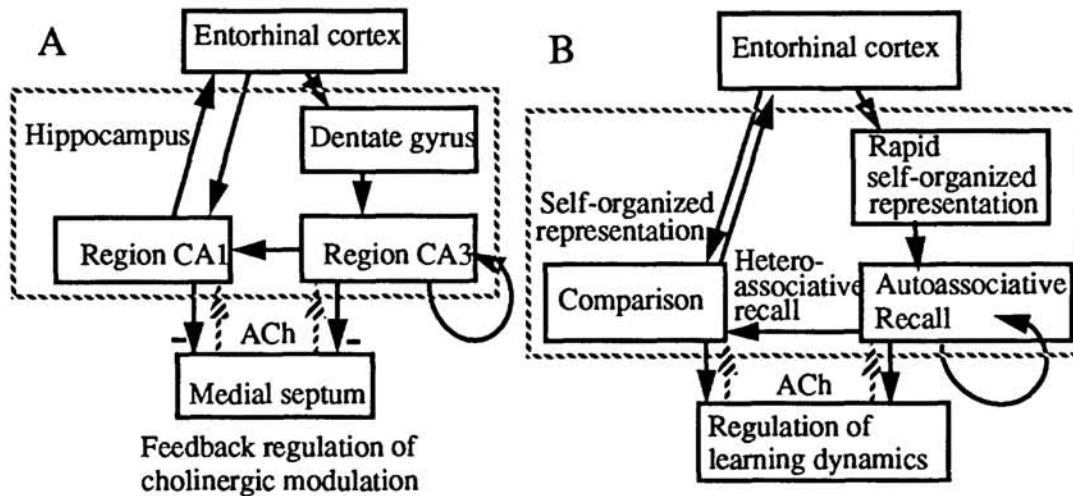

Figure 2. Schematic representation of hippocampal circuitry and the corresponding function of connections in the model.

## 2.3. CHOLINERGIC MODULATION

The total output from region CA1 determined the level of cholinergic modulation within both region CA3 and CA1, with increased output causing decreased modulation. This is consistent with experimental evidence suggesting that activity in region CA1 and region CA3 can inhibit activity in the medial septum, and thereby downregulate cholinergic modulation. This effect was obtained in the model by excitatory connections from region CA1 to an inhibitory interneuron in the medial septum, which suppressed the activity of a cholinergic neuron providing modulation to the full network. When levels of cholinergic modulation were high, there was strong suppression of synaptic transmission at the excitatory recurrent synapses in CA3 and the Schaffer collaterals projecting from region CA3 to CA1. This prevented the spread of activity due to previous learning from interfering with self-organization. When levels of cholinergic modulation were decreased, the strength of synaptic transmission was increased, allowing associative recall to dominate. Cholinergic modulation also increased the rate of synaptic modification and depolarized neurons.

## 2.4. TESTS OF SELF-REGULATED LEARNING AND RECALL

Simulations of the full hippocampal network evaluated the response to the sequential presentation of a series of highly overlapping activity patterns in the entorhinal cortex. Recall was tested with interspersed presentation of degraded versions of previously presented activity patterns. For effective recall, the pattern of activity in entorhinal cortex layer IV evoked by degraded patterns matched the pattern evoked by the full learned version of these patterns. The function of the full network is illustrated in Figure 3. In simulations

focused on region CA3, activity patterns were induced sequentially in region CA3, representing afferent input from the entorhinal cortex. Different levels of external activation of the cholinergic neuron resulted in different levels of learning of new overlapping patterns. These results are illustrated in Figure 4.

## 2.5. BRAIN SLICE EXPERIMENTS

The effects in the simulations of region CA3 depended upon the cholinergic suppression of synaptic transmission in stratum radiatum of this region  The cholinergic suppression of glutamatergic synaptic transmission in region CA3 was tested in brain slice preparations by analysis of the influence of the cholinergic agonist carbachol on the size of field potentials elicited by stimulation of stratum radiatum.  These experiments used techniques similar to previously published work in region CA1 (Hasselmo and Schnell, 1994).

## 3   RESULTS

In the full hippocampal simulation, input of an unfamiliar pattern to entorhinal cortex layer II resulted in high levels of acetylcholine.  This allowed rapid self-organization of the perforant path input to the dentate gyrus and region CA1.  Cholinergic suppression of synaptic transmission in region CA1 prevented recall from interfering with self-organization.  Instead, recurrent collaterals in region CA3 stored an autoassociative representation of the input from the dentate gyrus to region CA3, and connections from CA3 to CA1 stored associations between the pattern of activity in CA3 and the associated self-organized representation in region CA1.

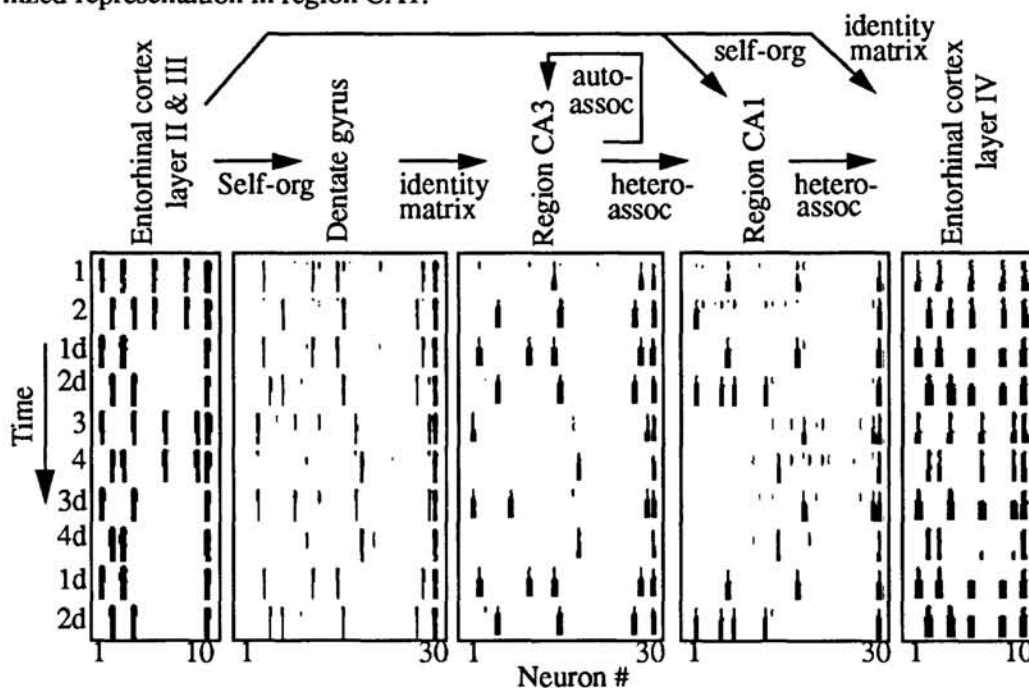

Figure 3. Activity in each subregion of the full network simulation of the hippocampal formation during presentation of a sequence of activity patterns in entorhinal cortex.

In Figure 3, width of the lines represents the activity of each neuron at a particular time step. As seen here, the network forms a self-organized representation of each new pattern consisting of active neurons in the dentate gyrus and region CA1. At the same time, an association is formed between the self-organized representation in region CA1 and the same afferent input pattern presented to entorhinal cortex layer IV. Four overlapping patterns (1-4) are presented sequentially, each of which results in learning of a separate self-organized representation in the dentate gyrus and region CA1, with an association formed between this representation and the full input pattern in entorhinal cortex.

The recall characteristics of the network are apparent when degraded versions of the afferent input patterns are presented in the sequence (1d-4d). This degraded afferent input weakly activates the same representations previously formed in the dentate gyrus. Recurrent excitation in region CA3 enhances this activity, giving robust recall of the full version of this pattern. This activity then reaches CA1, where it causes strong activation if it matches the pattern of afferent input from the entorhinal cortex. Strong activation in region CA1 decreases cholinergic modulation, preventing formation of a new representation and allowing recall to dominate. Strong activation of the representation stored in region CA1 then activates the full representation of the pattern in entorhinal cortex layer IV. Thus, the network can accurately recall each of many highly overlapping patterns.

The effect of cholinergic modulation on the level of learning or recall can be seen more clearly in a simulation of auto-associative memory function in region CA3 as shown in Figure 4. Each box shows the response of the network to sequential presentation of full and degraded versions of two highly overlapping input patterns. The width of the black traces represents the activity of each of 10 CA3 pyramidal cells during each simulation step. In the top row, level of cholinergic modulation (ACh) is plotted. In A, external activation of the cholinergic neuron is absent, so there is no cholinergic suppression of synaptic transmission. In this case, the first pattern is learned and recalled properly, but subsequent presentation of a second overlapping pattern results only in recall of the previously learned pattern. In B, with greater cholinergic suppression, recall is suppressed sufficiently to allow learning of a combination of the two input patterns. Finally, in C, strong cholinergic suppression prevents recall, allowing learning of the new overlapping pattern to dominate over the previously stored pattern.

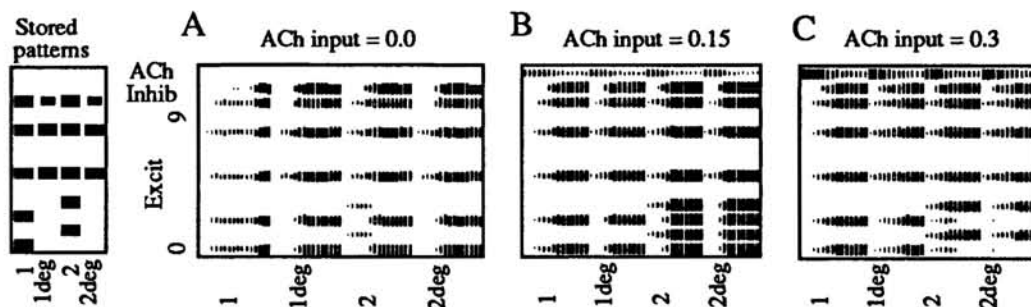

Figure 4. Increased cholinergic suppression of synaptic transmission in region CA3 causes greater learning of new aspects of afferent input patterns.

Extracellular recording in brain slice preparations of hippocampal region CA3 have demonstrated that perfusion of the cholinergic agonist carbachol strongly suppresses synaptic potentials recorded in stratum radiatum, as shown in Figure 5. In contrast, suppression of synaptic transmission at the afferent fiber synapses arising from entorhinal cortex is much weaker. At a concentration of 20μM, carbachol suppressed synaptic potentials in stratum radiatum on average by 54.4% (n=5). Synaptic potentials elicited in stratum lacunosum were more weakly suppressed, with an average suppression of 28%.

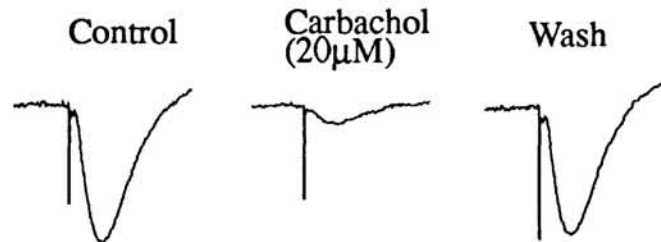

Figure 5. Cholinergic suppression of synaptic transmission in stratum radiatum of CA3.

## 4   DISCUSSION

In this model of the hippocampus, self-organization at perforant path synapses forms compressed representations of specific patterns of cortical activity associated with events in the environment. Feedback regulation of cholinergic modulation sets appropriate dynamics for learning in response to novel stimuli, allowing predominance of self-organization, and appropriate dynamics for recall in response to familiar stimuli, allowing predominance of associative memory function. This combination of self-organization and associative memory function may also occur in neocortical structures. The selective cholinergic suppression of feedback and intrinsic synapses has been proposed to allow self-organization of feedforward synapses while feedback synapses mediate storage of associations between higher level representations and activity in primary cortical areas (Hasselmo, 1994b). This previous proposal could provide a physiological justification for a similar mechanism utilized in recent models (Dayan et al., 1995). Detailed modeling of cholinergic effects in the hippocampus provides a theoretical framework for linking the considerable behavioral evidence for a role of acetylcholine in memory function (Hagan and Morris, 1989) to the neurophysiological evidence for the effects of acetylcholine within cortical structures (Hasselmo and Bower, 1992; 1993; Hasselmo, 1994a, 1994b).

**Acknowledgements**

This work supported by a pilot grant from the Massachusetts Alzheimer's Disease Research Center and by an NIMH FIRST award MH52732-01.

**References**

Barkai E, Hasselmo ME (1994) Modulation of the input/output function of rat piriform cortex pyramidal cells. *J. Neurophysiol.* 72: 644-658.

Barkai E, Bergman RE, Horwitz G, Hasselmo ME (1994) Modulation of associative memory function in a biophysical simulation of rat piriform cortex. *J. Neurophysiol.* 72:659-677.

Burgess N, Recce M, O'Keefe J (1994) A model of hippocampal function. *Neural Networks* 7: 1065-1081.

Dayan P, Hinton GE, Neal RM and Zemel RS (1995) The Helmholtz machine. *Neural computation* in press.

Eichenbaum, H. and Buckingham, J. (1990) Studies on hippocampal processing: experiment, theory and model.  In: Learning and computational neuroscience: foundations of adaptive networks, M. Gabriel and J. Moore, eds., Cambridge, MA: MIT Press.

Hagan, JJ and Morris, RGM (1989) The cholinergic hypothesis of memory: A review of animal experiments. In *Psychopharmacology of the Aging Nervous System*, L.L. Iversen, S.D. Iversen and S.H. Snyder, eds. New York: Plenum Press, p. 237-324.

Hasselmo, M.E. (1993) Acetylcholine and learning in a cortical associative memory. *Neural Comp.* 5: 22-34.

Hasselmo ME (1994a) Runaway synaptic modification in models of cortex: Implications for Alzheimer's disease. *Neural Networks* 7: 13-40.

Hasselmo ME (1994b) Neuromodulation and cortical function. *Behav. Brain Res.* in press

Hasselmo ME, Anderson, BP and Bower, JM (1992) Cholinergic modulation of cortical associative memory function. *J. Neurophysiol.* 67(5): 1230-1246.

Hasselmo ME, Bower JM (1992) Cholinergic suppression specific to intrinsic not afferent fiber synapses in rat piriform (olfactory) cortex. *J. Neurophysiol.* 67(5): 1222-1229.

Hasselmo ME, Bower JM (1993) Acetylcholine and memory. *Trends Neurosci* 16:218-222.

Hasselmo ME, Barkai E, Horwitz G, Bergman RE (1993) Modulation of neuronal adaptation and cortical associative memory function. In: Computation and Neural Systems II (Eeckman F, Bower JM, ed). Norwell, MA: Kluwer Academic Publishers.

Hasselmo ME, Schnell E (1994) Laminar selectivity of the cholinergic suppression of synaptic transmission in rat hippocampal region CA1: Computational modeling and brain slice physiology. *J. Neurosci.* 14: 3898-3914.

Levy WB (1989) A computational approach to hippocampal function. In: Computational models of learning in simple neural systems (Hawkins RD, Bower GH, ed), pp. 243-305. Orlando, FL: Academic Press.

Myers CE and Gluck M (1994) Context, conditioning and hippocampal rerepresentation in animal learning. *Behav. Neurosci.* 108: 835-847.

Marr D (1971) Simple memory: A theory for archicortex. *Phil. Trans. Roy. Soc. B* B262:23-81

McNaughton BL (1991) Associative pattern completion in hippocampal circuits: New evidence and new questions. *Brain Res. Rev.* 16:193-220.

McNaughton BL, Morris RGM (1987) Hippocampal synaptic enhancement and information storage within a distributed memory system. *Trends Neurosci.* 10:408-415.

Touretzky DS, Wan HS and Redish AD (1994) Neural representation of space in rats and robots.  In Zurada JM and Marks RJ (eds) Computational Intelligence: Imitating life. IEEE Press.
